# Heavy-tailed Distances for Gradient Based Image Descriptors

**Yangqing Jia and Trevor Darrell**
UC Berkeley EECS and ICSI
{jiayq,trevor}@eecs.berkeley.edu

## Abstract

Many applications in computer vision measure the similarity between images or image patches based on some statistics such as oriented gradients. These are often modeled implicitly or explicitly with a Gaussian noise assumption, leading to the use of the Euclidean distance when comparing image descriptors. In this paper, we show that the statistics of gradient based image descriptors often follow a heavy-tailed distribution, which undermines any principled motivation for the use of Euclidean distances. We advocate for the use of a distance measure based on the likelihood ratio test with appropriate probabilistic models that fit the empirical data distribution. We instantiate this similarity measure with the Gamma-compound-Laplace distribution, and show significant improvement over existing distance measures in the application of SIFT feature matching, at relatively low computational cost.

## 1   Introduction

A particularly effective image representation has developed in recent years, formed by computing the statistics of oriented gradients quantized into various spatial and orientation selective bins. SIFT [14], HOG [6], and GIST [17] have been shown to have extraordinary descriptiveness on both instance and category recognition tasks, and have been designed with invariances to many common nuisance parameters. Significant motivation for these architectures arises from biology, where models of early visual processing similarly integrate statistics over orientation selective units [21, 18].

Two camps have developed in recent years regarding how such descriptors should be compared. The first advocates comparison of raw descriptors. Early works [6] considered the distance of patches to a database from labeled images; this idea was reformulated as a probabilistic classifier in the NBNN technique [4], which has surprisingly strong performance across a range of conditions. Efficient approximations based on hashing [22, 12] or tree-based data structures [14, 16] or their combination [19] have been commonly applied, but do not change the underlying ideal distance measure.

The other approach is perhaps the more dominant contemporary paradigm, and explores a quantized-prototype approach where descriptors are characterized in terms of the closest prototype, e.g., in a vector quantization scheme. Recently, hard quantization and/or Euclidean-based reconstruction techniques have been shown inferior to sparse coding methods, which employ a sparsity prior to form a dictionary of prototypes. A series of recent publications has proposed prototype formation methods including various sparsity-inducing priors, including most commonly the $L_1$ prior [15], as well as schemes for sharing structure in a ensemble-sparse fashion across tasks or conditions [10]. It is informative that sparse coding methods also have a foundation as models for computational visual neuroscience [18].

Virtually all these methods use the Euclidean distance when comparing image descriptors against the prototypes or the reconstructions, which is implicitly or explicitly derived from a Gaussian noise assumption on image descriptors. In this paper, we ask whether this is the case, and further, whether

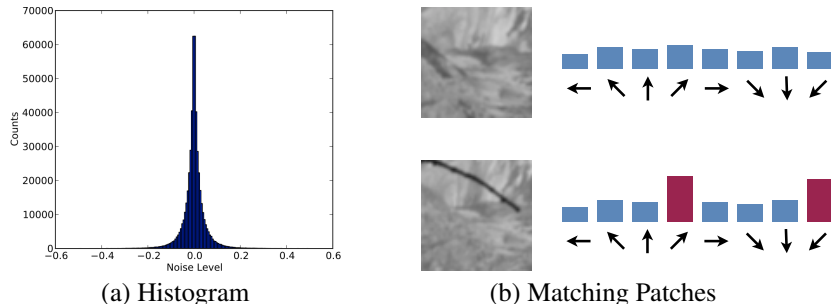

|                                     |                                 |
| :---------------------------------: | :-----------------------------: |
| (a) Histogram                       | (b) Matching Patches            |

Figure 1: (a) The histogram of the difference between SIFT features of matching image patches from the Photo Tourism dataset. (b) A typical example of matching patches. The obstruction (wooden branch) in the bottom patch leads to a sparse change to the histogram of oriented gradients (the two red bars).

there is a distance measure that better fits the distribution of real-world image descriptors. We begin by investigating the statistics of oriented gradient based descriptors, focusing on the well known Photo Tourism database [25] of SIFT descriptors for the case of simplicity. We evaluate the statistics of corresponding patches, and see the distribution is heavy-tailed and decidedly non-Gaussian, undermining any principled motivation for the use of Euclidean distances.

We consider generative factors why this may be so, and derive a heavy-tailed distribution (that we call the gamma-compound-Laplace distribution) in a Bayesian fashion, which empirically fits well to gradient based descriptors. Based on this, we propose to use a principled approach using the likelihood ratio test to measure the similarity between data points under any arbitrary parameterized distribution, which includes the previously adopted Gaussian and exponential family distributions as special cases. In particular, we prove that for the heavy-tailed distribution we proposed, the corresponding similarity measure leads to a distance metric, theoretically justifying its use as a similarity measurement between image patches.

The contribution of this paper is two-fold. We believe ours is the first work to systematically examine the distribution of the noise in terms of oriented gradients for corresponding keypoints in natural scenes. In addition, the likelihood ratio distance measure establishes a principled connection between the distribution of data and various distance measures in general, allowing us to choose the appropriate distance measure that corresponds to the true underlying distribution in an application. Our method serves as a building block in either nearest-neighbor distance computation (e.g. NBNN [4]) and codebook learning (e.g. vector quantization and sparse coding), where the Euclidean distance measure can be replaced by our distance measure for better performance.

It is important to note that in both paradigms listed above – nearest-neighbor distance computation and codebook learning – discriminative variants and structured approaches exist that can optimize a distance measure or codebook based on a given task. Learning a distance measure that incorporate both the data distribution and task-dependent information is the subject of future work.

## 2 Statistics of Local Image Descriptors

In this section, we focus on examining the statistics of local image descriptors, using the SIFT feature [14] as an example. Classical feature matching and clustering methods on SIFT features use the Euclidean distance to compare two descriptors. In a probabilistic perspective, this implies a Gaussian noise model for SIFT: given a feature prototype $\mu$ (which could be the prototype in feature matching, or a cluster center in clustering), the probability that an observation $x$ matches the prototype can be evaluated by the Gaussian probability

$$p(x|\mu) \propto \exp\left(\frac{\|x - \mu\|_2^2}{2\sigma^2}\right),\qquad(1)$$

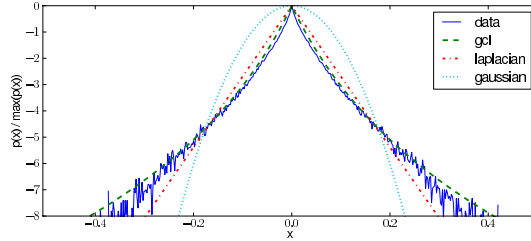

Figure 2: The probability values of the GCL, Laplace and Gaussian distributions via ML estimation, compared against the empirical distribution of local image descriptor noises. The figure is in log scale and curves are normalized for better comparison. For details about the data, see Section 4.

where $\sigma$ is the standard deviation of the noise. Such a Gaussian noise model has been explicitly or implicitly assumed in most algorithms including vector quantization, sparse coding (on the reconstruction error), etc.

Despite the popular use of Euclidean distance, the distribution of the noise between matching SIFT patches does not follow a Gaussian distribution: as shown in Figure 1(a), the distribution is highly kurtotic and heavy tailed, indicating that Euclidean distance may not be ideal.

The reason why the Gaussian distribution may not be a good model for the noise of local image descriptors can be better understood from the generative procedure of the SIFT features. Figure 1(b) shows a typical case of matching patches: one patch contains a partially obstructing object while the other does not. The resulting histogram differs only in a *sparse subset* of the oriented gradients. Further, research on the V1 receptive field [18] suggests that natural images are formed from localized, oriented, bandpass patterns, implying that changing the weight of one such building pattern may tend to change only one or a few dimensions of the binned oriented gradients, instead of imposing an isometric Gaussian change to the whole feature.

## 2.1    A Heavy-tailed Distribution for Image Descriptors

We first explore distributions that fits such heavy-tailed property. A common approach to cope with heavy-tails is to use the $L_1$ distance, which corresponds to the Laplace distribution

$$p(x|\mu;\lambda) \propto \frac{\lambda}{2} \exp\left(-\lambda|x-\mu|\right). \tag{2}$$

However, the tail of the noise distribution is often still heavier than the Laplace distribution: empirically, we find the kurtosis of the SIFT noise distribution to be larger than 7 for most dimensions, while the kurtosis of the Laplace distribution is only 3. Inspired by the hierarchical Bayesian models [11], instead of fixing the $\lambda$ value in the Laplace distribution, we introduce a conjugate Gamma prior over $\lambda$ modeled by hyperparameters $\{\alpha, \beta\}$, and compute the probability of $x$ given the prototype $\mu$ by integrating over $\lambda$:

$$
\begin{aligned}
p(x|\mu;\alpha,\beta) &= \int_\lambda \frac{\lambda}{2} e^{-\lambda|x-\mu|} \frac{1}{\Gamma(\alpha)} \lambda^{\alpha-1} \beta^\alpha e^{-\beta\lambda} \, \mathrm{d}\lambda \\
&= \frac{1}{2} \alpha\beta^\alpha (|x-\mu|+\beta)^{-\alpha-1}.
\end{aligned}
\tag{3}
$$

This leads to a heavier tail than the Laplace distribution. We call Equation (3) the Gamma-compound-Laplace (GCL) distribution, in which the hyperparameters $\alpha$ and $\beta$ control the shape of the tail. Figure 2 shows the empirical distribution of the SIFT noise and the maximum likelihood fitting of various models. It can be observed that the GCL distribution enables us to fit the heavy tailed empirical distribution better than other distributions. We note that similar approaches have been exploited in the compressive sensing context [9], and are shown to perform better than using the Laplace distribution as the sparse prior in applications such as signal recovery.

Further, we note that the statistics of a wide range of other natural image descriptors beyond SIFT features are known to be highly non-Gaussian and have heavy tails [24]. Examples of these include

derivative-like wavelet filter responses [23, 20], optical flow and stereo vision statistics [20, 8], shape from shading [3], and so on.

In this paper we retract from the general question "what is the right distribution for natural images", and ask specifically whether there is a good distance metric for local image descriptors that takes the heavy-tailed distribution into consideration. Although heuristic approaches such as taking the squared root of the feature values before computing the Euclidean distance are sometimes adopted to alleviate the effect of heavy tails, there lacks a principled way to define a distance for heavy-tailed data in computer vision to the best of our knowledge. To this end, we start with a principled similarity measure based on the well known statistical hypothesis test, and instantiate it with heavy-tailed distributions we propose for local image descriptors.

## 3  Distance For Heavy-tailed Distributions

In statistics, the hypothesis test [7] approach has been widely adopted to test if a certain statistical model fits the observation. We will focus on the likelihood ratio test in this paper. In general, we assume that the data is generated by a parameterized probability distribution $p(x|\boldsymbol{\theta})$, where $\boldsymbol{\theta}$ is the vector of parameters. A null hypothesis is stated by restricting the parameter $\boldsymbol{\theta}$ in a specific subset $\boldsymbol{\Theta}_0$, which is nested in a more general parameter space $\boldsymbol{\Theta}$. To test if the restricted null hypothesis fits a set of observations $\mathcal{X}$, a natural choice is to use the ratio of the maximized likelihood of the restricted model to the more general model:

$$\Lambda(\mathcal{X}) = L(\hat{\boldsymbol{\theta}}_0; \mathcal{X})/L(\hat{\boldsymbol{\theta}}; \mathcal{X}), \tag{4}$$

where $L(\theta; \mathcal{X})$ is the likelihood function, $\hat{\boldsymbol{\theta}}_0$ is the maximum likelihood estimate of the parameter within the restricted subset $\boldsymbol{\Theta}_0$, and $\hat{\boldsymbol{\theta}}$ is the maximum likelihood estimate under the general case.

It is easily verifiable that $\Lambda(\mathcal{X})$ always lies in the range $[0, 1]$, as the maximum likelihood estimate of the general case would always fit at least as well as the restricted case, and that the likelihood is always a nonnegative value. The *likelihood ratio test* is then defined as a statistical test that rejects the null hypothesis when the statistic $\Lambda(\mathcal{X})$ is smaller than a certain threshold $\alpha$, such as the Pearson's chi-square test [7] for categorical data.

Instead of producing a binary decision, we propose to use the score directly as the generative similarity measure between two single data points. Specifically, we assume that each data point $x$ is generated from a parameterized distribution $p(x|\mu)$ with unknown prototype $\mu$. Thus, the statement "two data points $x$ and $y$ are similar" can be reasonably represented by the null hypothesis that the two data points are generated from the same prototype $\mu$, leading to the probability

$$q_0(x, y|\mu_{xy}) = p(x|\mu_{xy})p(y|\mu_{xy}). \tag{5}$$

This restricted model is further nested in the more general model that generates the two data points from two possibly different prototypes:

$$q(x, y|\mu_x, \mu_y) = p(x|\mu_x)p(y|\mu_y), \tag{6}$$

where $\mu_x$ and $\mu_y$ are not necessarily equal.

The similarity between the two data points $x$ and $y$ is then defined by the the likelihood ratio statistics between the null hypothesis of equality and the alternate hypothesis of inequality over prototypes:

$$s(x, y) = \frac{p(x|\hat{\mu}_{xy})p(y|\hat{\mu}_{xy})}{p(x|\hat{\mu}_x)p(y|\hat{\mu}_y)}, \tag{7}$$

where $\hat{\mu}_x$, $\hat{\mu}_y$ and $\hat{\mu}_{xy}$ are the maximum likelihood estimates of the prototype based on $x$, $y$, and $\{x, y\}$ respectively. We call (7) the *likelihood ratio similarity* between $x$ and $y$, which provides us information from a generative perspective: two similar data points, such as two patches of the same real-world location, are more likely to be generated from the same underlying distribution, thus have a large likelihood ratio value. In the following parts of the paper, we define the *likelihood ratio distance* between $x$ and $y$ as the square root of the negative logarithm of the similarity:

$$d(x, y) = \sqrt{-\log(s(x, y))}. \tag{8}$$

It is worth pointing out that, for arbitrary distributions $p(x)$, $d(x, y)$ is not necessarily a distance metric as the triangular inequality may not hold. However, for heavy-tailed distributions, we have the following sufficient condition in the 1-dimensional case:

**Theorem 3.1.** *If the distribution $p(x|\mu)$ can be written as $p(x|\mu) = \exp(-f(x-\mu))b(x)$, where $f(t)$ is a non-constant quasiconvex function w.r.t. $t$ that satisfies $f''(t) \leq 0$, $\forall t \in \mathbb{R}\backslash\{0\}$, then the distance defined in Equation (8) is a metric.*

*Proof.* First we point out the following lemmas:

**Lemma 3.2.** *If a function $d(x,y)$ defined on $\mathbb{X} \times \mathbb{X} \to \mathbb{R}$ is a distance metric, then $\sqrt{d(x,y)}$ is also a distance metric.*

**Lemma 3.3.** *If function $f(t)$ is defined as in Theorem 3.1, then we have:*
*(1) the minimizer $\hat{\mu}_{xy} = \arg\min_\mu f(x-\mu) + f(y-\mu)$ is either $x$ or $y$.*
*(2) the function $g(t) = \min(f(t), f(-t)) - f(0)$ is monotonically increasing and concave in $\mathbb{R}^+ \cup \{0\}$, and $g(0) = 0$.*

With Lemma 3.3, it is easily verifiable that $d^2(x,y) = g(|x-y|)$. Then, via the subadditivity of $g(\cdot)$ we can reach a result stronger than Theorem 3.1 that $d^2(x,y)$ is a distance metric. Thus, $d(x,y)$ is also a distance metric based on Lemma 3.2. Note that we keep the square root here in conformity with classical distance metrics, which we will discuss in the later parts of the paper. Detailed proofs of the theorem and lemmas can be found in the supplementary material. $\square$

As an extreme case, when $f''(t) = 0$ ($t \neq 0$), the distance defined above is the square root of the (scaled) $L_1$ distance.

## 3.1 Distance for the GCL distribution

We use the GCL distribution parameterized by the prototype $\mu$ with fixed hyperparameters $(\alpha, \beta)$ as the SIFT noise model, which leads to the following GCL distance between dimensions of SIFT patches[1]:

$$d^2(x,y) = (\alpha+1)(\log(|x-y|+\beta) - \log\beta) \tag{9}$$

The distance between two patches is then defined as the sum of per-dimension distances. Intuitively, while the Euclidean distance grows linearly w.r.t. to the difference between the coordinates, the GCL distance grows in a logarithmic way, suppressing the effect of too large differences. Further, we have the following theoretical justification which is a direct result of Theorem 3.1.:

**Proposition 3.4.** *The distance $d(x,y)$ defined in (9) is a metric.*

## 3.2 Hyperparameter Estimation for GCL

In the following, we discuss how to estimate the hyperparameters $\alpha$ and $\beta$ in the GCL distribution. Assuming that we are given a set of one-dimensional data $\mathcal{D} = \{x_1, x_2, \cdots, x_n\}$ that follows the GCL distribution, we estimate the hyperparameters by maximizing the log likelihood

$$l(\alpha, \beta; \mathcal{D}) = \sum_{i=1}^n \left( \log\frac{\alpha}{2} + \alpha\log\beta - (\alpha+1)\log(|x_i|+\beta) \right) \tag{10}$$

The ML estimation does not have a closed-form solution, so we adopt an alternate optimization and iteratively update $\alpha$ and $\beta$ until convergence. Updating $\alpha$ with fixed $\beta$ can be achieved by computing

$$\alpha \leftarrow n \left( \sum_{i=1}^n \log(|x_i|+\beta) - n\log(\beta) \right)^{-1}. \tag{11}$$

Updating $\beta$ can be done via the Newton-Raphson method $\beta \leftarrow \beta - \frac{l'(\beta)}{l''(\beta)}$, where

$$l'(\beta) = \frac{n\alpha}{\beta} - \sum_{i=1}^n \frac{\alpha+1}{|x_i|+\beta}, \quad l''(\beta) = \sum_{i=1}^n \frac{\alpha+1}{(|x_i|+\beta)^2} - \frac{n\alpha}{\beta^2} \tag{12}$$

### 3.3 Relation to Existing Measures

The likelihood ratio distance is related to several existing methods. In particular, we show that under the exponential family distribution, it leads to several widely used distance measures.

The exponential family distribution has drawn much attention in the recent years. Here we focus on the regular exponential family, where the distribution of data $x$ can be written in the following form:

$$p(x) = \exp\left(-d_B(x, \mu)\right) b(x), \tag{13}$$

where $\mu$ is the mean in the exponential family sense, and $d_B$ is the regular Bregman divergence corresponding to the distribution [2]. When applying the likelihood ratio distance on the distribution, we obtain the distance

$$d(x, y) = \sqrt{d_B(x, \hat{\mu}_{xy}) + d_B(x, \hat{\mu}_{x,y})} \tag{14}$$

since $\hat{\mu}_x \equiv x$ and $d_B(x, x) \equiv 0$ for any $x$. We note that this is the square root of the Jensen-Bregman divergence and is known to be a distance metric [1]. Several popular distances can be derived in this way. In the two most common cases, the Gaussian distribution leads to the Euclidean distance, and the multinomial distribution leads to the square root of the Jensen-Shannon divergence, whose first-order approximation is the $\chi$-squared distance. More generally, for (non-regular) Bregman divergences $d_B(x, \mu)$ defined as $d_B(x, \mu) = F(x) - F(\mu) + (x - \mu)F'(\mu)$ with arbitrary smooth function $F$, the condition on which the square root of the corresponding Jensen-Bregman divergence is a metric has been discussed in [5].

While the exponential family embraces a set of mathematically elegant distributions whose properties are well known, it fails to capture the heavy-tailed property of various natural image statistics, as the tail of the sufficient statistics is exponentially bounded by definition. The likelihood ratio distance with heavy-tailed distributions serves as a principled extension of several popular distance metrics based on the exponential family distribution. Further, there are principled approaches that connect distances with kernels [1], upon which kernel methods such as support vector machines may be built with possible heavy-tailed property of the data taken into consideration.

The idea of computing the similarity between data points based on certain scores has also been seen in the one-shot learning context [26] that uses the average prediction score taking one data point as training and the other as testing, and vice versa. Our method shares similar merit, but with a generative probabilistic interpretation. Integration of our method with discriminative information or latent application-dependent structures is one future direction.

## 4 Experiments

In this section, we apply the GCL distance to the problem of local image patch similarity measure using the SIFT feature, a common building block of many applications such as stereo vision, structure from motion, photo tourism, and bag-of-words image classification.

### 4.1 The Photo Tourism Dataset

We used the Photo Tourism dataset [25] to evaluate different similarity measures of the SIFT feature. The dataset contains local image patches extracted from three scenes namely `Notredame`, `Trevi` and `Halfdome`, reflecting different natural scenarios. Each set contains approximately 30,000 ground-truth 3D points, with each point containing a bag of 2d image patches of size $64 \times 64$ corresponding to the 3D point. To the best of our knowledge, this is the largest local image patch database with ground-truth correspondences. Figure 3 shows a typical subset of patches from the dataset.

The SIFT features are computed using the code in [13]. Specifically, two different normalization schemes are tested: the *l2* scheme simply normalizes each feature to be of length 1, and the *thres* scheme further thresholds the histogram at $0.2$, and rescales the resulting feature to length 1. The latter is the classical hand-tuned normalization designed in the original SIFT paper, and can be seen as a heuristic approach to suppress the effect of heavy tails.

Following the experimental setting of [25], we also introduce random jitter effects to the raw patches before SIFT feature extraction by warping each image by the following random warping parame-

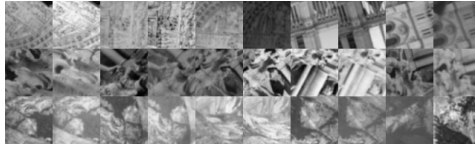

Figure 3: An example of the Photo Tourism dataset. From top to bottom patches are sampled from Notredame, Trevi and Halfdome respectively. Within each row, every adjacent two patches forms a matching pair.

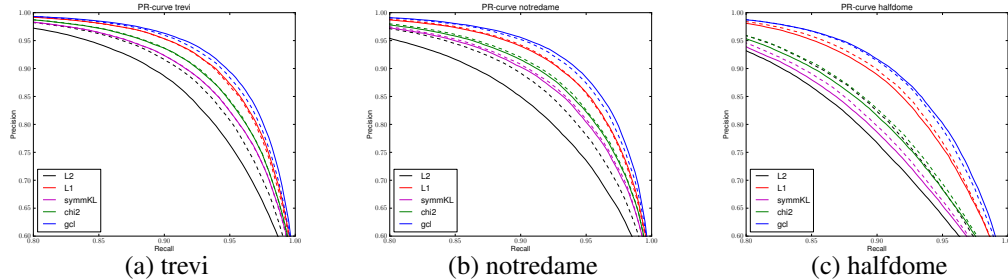

(a) trevi        (b) notredame        (c) halfdome

Figure 4: The mean precision-recall curve over 20 independent runs. In the figure, solid lines are experiments using features that are normalized in the *l2* scheme, and dashed lines using features normalized in the *thres* scheme. Best viewed in color.

ters: position shift, rotation and scale with standard deviations of 0.4 pixels, 11 degrees and 0.12 octaves respectively. Such jitter effects represent the noise we may encounter in real feature detection and localization [25], and allows us to test the robustness of different distance measures. For completeness, the data without jitter effects are also tested and the results reported.

## 4.2 Testing Protocol

The testing protocol is as follows: 10,000 matching pairs and 10,000 non-matching pairs are randomly sampled from the dataset, and we classify each pair to be matching or non-matching based on the distance computed from different testing metrics. The precision-recall (PR) curve is computed, and two values, namely the average precision (AP) computed as the area under the PR curve and the false positive rate at 95% recall (95%-FPR) are reported to compare different distance measures. To test the statistical significance, we carry out 20 independent runs and report the mean and standard deviation in the paper. We focus on comparing distance measures that presume the data to lie in a vector space. Five different distance measures are compared, namely the $L_2$ distance, the $L_1$ distance, the symmetrized KL divergence, the $\chi^2$ distance, and the GCL distance.

The hyperparameters of the GCL distance measure are learned by randomly sampling 50,000 matching pairs from the set `Notredame`, and performing hyperparameter estimation as described in Section 3.2. They are then fixed and used universally for all other experiments without re-estimation. As a final note, the code for the experiments in the paper will be released to public for repeatability.

## 4.3 Experimental Results

Figure 4 shows the average precision-recall curve for all the distances on the three datasets respectively. The numerical results on the data with jitter effects are summarized in Table 1, with statistically significant values shown in bold. Table 2 shows the 99% FPR on the data without jitter effects[2]. We refer to the supplementary materials for other results on the no jitter case due to space constraints. Notice that, the observed trends and conclusions from the experiments with jitter effects are also confirmed on those without jitter effects.

The GCL distance outperforms other base distance measures in all the experiments. Notice that the hyperparameters learned from the `notredame` set performs well on the other two datasets as well,

| AP | $L_2$ | $L_1$ | SymmKL | $\chi^2$ | GCL |
|---|---|---|---|---|---|
| trevi-l2 | 96.61±0.16 | 98.08±0.10 | 97.40±0.12 | 97.69±0.11 | **98.33±0.09** |
| trevi-thres | 97.23±0.12 | 98.05±0.10 | 97.40±0.11 | 97.71±0.11 | **98.21±0.10** |
| notre-l2 | 95.90±0.14 | 97.83±0.10 | 96.96±0.12 | 97.31±0.11 | **98.19±0.10** |
| notre-thres | 96.76±0.13 | 97.84±0.10 | 97.05±0.12 | 97.39±0.11 | **98.07±0.10** |
| halfd-l2 | 94.51±0.16 | 96.75±0.11 | 94.87±0.15 | 95.42±0.14 | **98.19±0.10** |
| halfd-thres | 95.55±0.14 | 96.90±0.11 | 95.08±0.16 | 95.64±0.14 | **97.21±0.10** |

| 95%-FPR | $L_2$ | $L_1$ | SymmKL | $\chi^2$ | GCL |
|---|---|---|---|---|---|
| trevi-l2 | 23.61±1.14 | 12.71±0.83 | 17.58±0.96 | 15.85±0.74 | **10.52±0.73** |
| trevi-thres | 19.23±0.84 | 13.08±0.91 | 17.57±0.98 | 15.66±0.77 | **11.21±0.71** |
| notre-l2 | 26.43±1.03 | 14.27±1.09 | 19.56±1.00 | 17.70±1.08 | **11.58±1.00** |
| notre-thres | 21.88±1.21 | 14.49±1.25 | 19.07±1.11 | 17.38±0.95 | **12.09±1.11** |
| halfd-l2 | 36.34±0.98 | 24.11±1.13 | 34.55±0.96 | 31.62±1.09 | **19.76±1.03** |
| halfd-thres | 31.44±1.20 | 23.14±0.13 | 33.71±1.05 | 30.56±1.13 | **20.74±1.16** |

Table 1: The average precision (above) and the false positive rate at 95% recall (below) of different distance measures on the Photo Tourism datasets, with random jitter effects. A larger AP score and a smaller FPR score are desired. The *l2* and *thres* in the leftmost column indicate the two different feature normalization schemes.

| 99%-FPR | $L_2$ | $L_1$ | SymmKL | $\chi^2$ | GCL |
|---|---|---|---|---|---|
| trevi-l2 | 11.36±1.65 | 3.44±0.75 | 8.02±1.04 | 8.02±1.08 | **2.42±0.58** |
| trevi-thres | 7.14±1.31 | 3.24±0.69 | 7.93±1.11 | 5.06±0.97 | **2.23±0.48** |
| notre-l2 | 19.69±1.93 | 6.09±0.72 | 14.81±1.66 | 9.40±1.04 | **4.16±0.57** |
| notre-thres | 11.9±1.19 | 5.17±0.58 | 13.11±1.39 | 8.24±1.12 | **3.72±0.56** |
| halfd-l2 | 44.55±9.42 | 34.01±2.10 | 43.51±1.07 | 40.53±1.12 | **26.06±2.25** |
| halfd-thres | 40.58±1.63 | 32.30±2.28 | 42.51±1.22 | 39.28±1.49 | **26.36±2.50** |

Table 2: The false positive rate at 99% recall of different distance measures on the Photo Tourism datasets without jitter effects.

indicating that they capture the general statistics of the SIFT feature, instead of dataset-dependent statistics. Also, the thresholding and renormalization of SIFT features does provide a significant improvement for the Euclidean distance, but its effect is less significant for other distances. In fact, the hard thresholding may introduce artificial noise to the data, counterbalancing the positive effect of reducing the tail, especially when the distance measure is already able to cope with heavy tails.

We argue that the key factor leading to the performance improvement is taking the heavy tail property of the data into consideration but not others. For instance, the Laplace distribution has a heavier tail than distributions corresponding to other base distance measures, and a better performance of the corresponding $L_1$ distance over other distance measures is observed, showing a positive correlation between tail heaviness and performance. Notice that the tails of distributions assumed by the baseline distances are still exponentially bounded, and performance is further increased by introducing heavy-tailed distributions such as the GCL distribution in our experiment.

## 5 Conclusion

While visual representations based on oriented gradients have been shown to be effective in many applications, scant attention has been paid to the issue of the heavy-tailed nature of their distributions, undermining the use of distance measures based on exponentially bounded distributions. In this paper, we advocate the use of distance measures that are derived from heavy-tailed distributions, where the derivation can be done in a principled manner using the log likelihood ratio test. In particular, we examine the distribution of local image descriptors, and propose the Gamma-compound-Laplace (GCL) distribution and the corresponding distance for image descriptor matching. Experimental results have shown that this yields to more accurate feature matching than existing baseline distance measures.

## Footnotes

[1]For more than two data points $\mathcal{X} = \{x_i\}$, it is generally difficult to find the maximum likelihood estimation of $\mu$ as the likelihood is nonconvex. However, with two data points $x$ and $y$, it is trivial to see that $\mu = x$ and $\mu = y$ are the two global optimums of the likelihood $L(\mu; \{x,y\})$, both leading to the same distance representation in (9).

[2]As the accuracy for the no jitter effects case is much higher in general, 99% FPR is reported instead of 95% FPR as in the jitter effect case.

# References

[1] A Agarwal and H Daume III. Generative kernels for exponential families. In *AISTATS*, 2011.

[2] A Banerjee, S Merugu, I Dhillon, and J Ghosh. Clustering with Bregman divergences. *JMLR*, 6:1705–1749, 2005.

[3] JT Barron and J Malik. High-frequency shape and albedo from shading using natural image statistics. In *CVPR*, 2011.

[4] O Boiman, E Shechtman, and M Irani. In defense of nearest-neighbor based image classification. In *CVPR*, 2008.

[5] P Chen, Y Chen, and M Rao. Metrics defined by bregman divergences. *Communications in Mathematical Sciences*, 6(4):915–926, 2008.

[6] N Dalal. Histograms of oriented gradients for human detection. In *CVPR*, 2005.

[7] AC Davison. *Statistical models*. Cambridge Univ Press, 2003.

[8] J Huang, AB Lee, and D Mumford. Statistics of range images. In *CVPR*, 2000.

[9] S Ji, Y Xue, and L Carin. Bayesian compressive sensing. *IEEE Trans. Signal Processing*, 56(6):2346–2356, 2008.

[10] Y Jia, M Salzmann, and D Trevor. Factorized latent spaces with structured sparsity. In *NIPS*, 2010.

[11] D Koller and N Friedman. *Probabilistic graphical models*. MIT press, 2009.

[12] B Kulis and T Darrell. Learning to hash with binary reconstructive embeddings. In *NIPS*, 2009.

[13] S Lazebnik, C Schmid, and J Ponce. Beyond bags of features: Spatial pyramid matching for recognizing natural scene categories. In *CVPR*, 2006.

[14] D Lowe. Distinctive image features from scale-invariant keypoints. *IJCV*, 60(2):91–110, 2004.

[15] J Mairal, F Bach, J Ponce, and G Sapiro. Online learning for matrix factorization and sparse coding. *JMLR*, 11:19–60, 2010.

[16] AW Moore. The anchors hierarchy: using the triangle inequality to survive high dimensional data. In *UAI*, 2000.

[17] A Oliva and A Torralba. Modeling the shape of the scene: A holistic representation of the spatial envelope. *IJCV*, 42(3):145–175, 2001.

[18] B Olshausen. Emergence of simple-cell receptive field properties by learning a sparse code for natural images. *Nature*, 381(6583):607–609, 1996.

[19] M Ozuysal and P Fua. Fast keypoint recognition in ten lines of code. In *CVPR*, 2007.

[20] J Portilla, V Strela, MJ Wainwright, and EP Simoncelli. Image denoising using scale mixtures of gaussians in the wavelet domain. *IEEE Trans. Image Processing*, 12(11):1338–1351, 2003.

[21] M Riesenhuber and T Poggio. Hierarchical models of object recognition in cortex. *Nature Neuroscience*, 2:1019–1025, 1999.

[22] G Shakhnarovich, P Viola, and T Darrell. Fast pose estimation with parameter-sensitive hashing. In *ICCV*, 2003.

[23] EP Simoncelli. Statistical models for images: compression, restoration and synthesis. In *Asilomar Conference on Signals, Systems & Computers*, 1997.

[24] Y Weiss and WT Freeman. What makes a good model of natural images? In *CVPR*, 2007.

[25] S Winder and M Brown. Learning local image descriptors. In *CVPR*, 2007.

[26] L Wolf, T Hassner, and Y Taigman. The one-shot similarity kernel. In *ICCV*, 2009.

